# ADAPTIVE KNOT PLACEMENT FOR NONPARAMETRIC REGRESSION

**Hossein L. Najafi***
Department of Computer Science
University of Wisconsin
River Falls, WI 54022

**Vladimir Cherkassky**
Department of Electrical Engineering
University of Minnesota
Minneapolis, Minnesota 55455

## Abstract

Performance of many nonparametric methods critically depends on the strategy for positioning knots along the regression surface. Constrained Topological Mapping algorithm is a novel method that achieves adaptive knot placement by using a neural network based on Kohonen's self-organizing maps. We present a modification to the original algorithm that provides knot placement according to the estimated second derivative of the regression surface.

## 1  INTRODUCTION

Here we consider regression problems. Using mathematical notation, we seek to find a function $f$ of $N - 1$ predictor variables (denoted by vector $\mathbf{X}$) from a given set of $n$ data points, or measurements, $\mathbf{Z}_i = (\mathbf{X}_i\ , Y_i)$ $(i = 1, \ldots, n)$ in $N$ - dimensional sample space:

$$Y = f(\mathbf{X}) + error \qquad (1)$$

where error is unknown (but zero mean) and its distribution may depend on $\mathbf{X}$. The distribution of points in the training set can be arbitrary, but uniform distribution in the domain of $\mathbf{X}$ is often used.

*Responsible for correspondence, Telephone (715) 425-3769, e-mail hossein.najafi@uwrf.edu.

The goal of this paper is to show how statistical considerations can be used to improve the performance of a novel neural network algorithm for regression [CN91], in order to achieve adaptive positioning of knots along the regression surface. By estimating and employing the second derivative of the underlying function, the modified algorithm is made more flexible around the regions with large second derivative. Through empirical investigation, we show that this modified algorithm allocates more units around the regions where the second derivative is large. This increase in the local knot density introduces more flexibility into the model (around the regions with large second derivative) and makes the model less biased around these regions. However, no over-fitting is observed around these regions.

## 2  THE PROBLEM OF KNOT LOCATION

One of the most challenging problems in practical implementations of adaptive methods for regression is adaptive positioning of knots along the regression surface. Typically, knot positions in the domain of $\mathbf{X}$ are chosen as a subset of the training data set, or knots are uniformly distributed in $\mathbf{X}$. Once $\mathbf{X}$-locations are fixed, commonly used data-driven methods can be applied to determine the number of knots. However, de Boor [dB78] showed that a polynomial spline with unequally spaced knots can approximate an arbitrary function much better than a spline with equally spaced knots. Unfortunately, the minimization problem involved in determination of the optimal placement of knots is highly nonlinear and the solution space is not convex [FS89]. Hence, the performance of many recent algorithms that include adaptive knot placement (e.g. MARS) is difficult to evaluate analytically. In addition, it is well-known that when data points are uniform, more knots should be located where the second derivative of the function is large. However, it is difficult to extend these results for non-uniform data in conjunction with data-dependent noise. Also, estimating the second derivative of a true function is necessary for optimal knot placement. Yet, the function itself is unknown and its estimation depends on the good placement of knots. This suggests the need for some iterative procedure that alternates between function estimation(smoothing) and knot positioning steps.

Many ANN methods effectively try to solve the problem of adaptive knot location using ad hoc strategies that are not statistically optimal. For example, local adaptive methods [Che92] are generalization of kernel smoothers where the kernel functions and kernel centers are determined from the data by some adaptive algorithm. Examples of local adaptive methods include several recently proposed ANN models known as radial basis function (RBF) networks, regularization networks, networks with locally tuned units etc [BL88, MD89, PG90]. When applied to regression problems, all these methods seek to find regression estimate in the (most general) form $\sum_{i=1}^{k} b_i \mathbf{H}_i(\mathbf{X}, C_i)$ where $\mathbf{X}$ is the vector of predictor variable, $C_i$ is the coordinates of the $i$-th 'center' or 'bump', $\mathbf{H}_i$ is the response function of the kernel type (the kernel width may be different for each center $i$), $b_i$ are linear coefficients to be determined, and $k$ is the total number of knots or 'centers'.

Whereas the general formulation above assumes global optimization of an error measure for the training set with respect to all parameters, i.e. center locations, kernel width and linear coefficients, this is not practically feasible because the error surface is generally non-convex and may have local minima [PG90, MD89]. Hence most

practical approaches first solve the problem of center(knot) location and assume identical kernel functions. Then the remaining problem of finding linear coefficients $b_i$ is solved by using familiar methods of Linear Algebra [PG90] or gradient-descent techniques [MD89]. It appears that the problem of center locations is the most critical one for the local neural network techniques. Unfortunately, heuristics used for center location are not based on any statistical considerations, and empirical results are too sketchy [PG90, MD89]. In statistical methods knot locations are typically viewed as free parameters of the model, and hence the number of knots directly controls the model complexity. Alternatively, one can impose local regularization constraints on adjacent knot locations, so that neighboring knots cannot move independently. Such an approach is effectively implemented in the model of self-organization known as Kohonen's Self-Organizing Maps (SOM) [Koh84]. This model uses a set of units ("knots") with neighborhood relations between units defined according to a fixed topological structure (typically 1D or 2D grid). During training or self-organization, data points are presented to the map iteratively, one at a time, and the unit closest to the data moves towards it, also pulling along its topological neighbors.

## 3  MODIFIED CTM ALGORITHM FOR ADAPTIVE KNOT PLACEMENT

The SOM model has been applied to nonparametric regression by Cherkassky and Najafi [CN91] in order to achieve adaptive positioning of knots along the regression surface. Their technique, called Constrained Topological Mapping (CTM), is a modification of Kohonen's self-organization suitable for regression problems. CTM interprets the units of the Kohonen map as movable knots of a regression surface. Correspondingly, the problem of finding regression estimate can be stated as the problem of forming an $M$ - dimensional topological map using a set of samples from $N$ - dimensional sample space (where $M \leq N - 1$) . Unfortunately, straightforward application of the Kohonen Algorithm to regression problem does not work well [CN91]. Because, the presence of noise in the training data can fool the algorithm to produce a map that is a multiple-valued function of independent variables in the regression problem (1). This problem is overcome in the CTM algorithm, where the nearest neighbor is found in the subspace of predictor variables, rather than in the input(sample) space [CN91].

We present next a concise description of the CTM algorithm. Using standard formulation (1) for regression, the training data are $N$ - dimensional vectors $\mathbf{Z}_i = (\mathbf{X}_i, Y_i)$, where $Yi$ is a noisy observation of an unknown function of $N - 1$ predictor variables given by vector $\mathbf{X}_i$. The CTM algorithm constructs an $M$ - dimensional topological map in $N$ - dimensional sample space ($M \leq N - 1$) as follows:

0. Initialize the $M$ - dimensional topological map in $N$ - dimensional sample space.

1. Given an input vector $\mathbf{Z}$ in $N$ - dimensional sample space, find the closest (best matching) unit $i$ in the subspace of independent variables:

$$\| \mathbf{Z}^*(k) - \mathbf{W}_i^* \| = Min_j\{\|\mathbf{Z}^* - \mathbf{W}_j^*\|\} \qquad \forall j \in [1,...,L]$$

where $\mathbf{Z}^*$ is the projection of the input vector onto the subspace of independent variables, $\mathbf{W}_j^*$ is the projection of the weight vector of unit j, and $k$ is the discrete time step.

2. Adjust the units' weights according to the following and return to 1:

$$\mathbf{W}_j(k+1) = W_j(k) + \beta(k)C_j(k)(\mathbf{Z}(k) - \mathbf{W}_j(k)) \qquad \forall j \qquad (2)$$

where $\beta(k)$ is the learning rate and $C_j(k)$ is the neighborhood for unit $j$ at iteration $k$ and are given by:

$$\beta(k) = \beta_0 \times \left(\frac{\beta_f}{\beta_0}\right)^{\left(\frac{k}{k_{max}}\right)} , C_j(k) = \frac{1}{\exp^{0.5\left(\frac{\|i-j\|}{\beta(k) \times S_0}\right)^2}} \qquad (3)$$

where $k_{max}$ is the final value of the time step ($k_{max}$ is equal to the product of the training set size by the number of times it was recycled), $\beta_0$ is the initial learning rate, and $\beta_f$ is the final learning rate ($\beta_0 = 1.0$ and $\beta_f = 0.05$ were used in all of our experiments), $\|i-j\|$ is the topological distance between the unit $j$ and the best matched unit $i$ and $S_0$ is the initial size of the map (i.e., the number of units per dimension) .

Note that CTM method achieves placement of units (knots) in X-space according to density of training data. This is due to the fact that X-coordinates of CTM units during training follow the standard Kohonen self-organization algorithm [Koh84], which is known to achieve faithful approximation of an unknown distribution. However, existing CTM method does not place more knots where the underlying function changes rapidly. The improved strategy for CTM knot placement in X-space takes into account estimated second derivative of a function as is described next.

The problem with estimating second derivative is that the function itself is unknown. This suggests using an iterative strategy for building a model, i.e., start with a crude model, estimate the second derivative based on this crude model, use the estimated second derivative to refine the model, etc. This strategy can be easily incorporated into the CTM algorithm due to its iterative nature. Specifically, in CTM method the map of knots(i.e., the model) becomes closer and closer to the final regression model as the training proceeds. Therefore, at each iteration, the modified algorithm estimates the second derivative at the best matching unit (closest to the presented data point in X-space), and allows additional movement of knots proportional to this estimate. Estimating the second derivative from the map (instead of using the training data) makes sense due to smoothing properties of CTM.

The modified CTM algorithm can be summarized as follows:

1. Present training sample $\mathbf{Z}_i = (\mathbf{X}_i , Y_i)$ to the map and find the closest (best matching) unit $i$ in the subspace of independent variables to this data point. (same as in the original CTM)

2. Move the the map (i.e., the best matching unit and all its neighbors) toward the presented data point (same as in the original CTM)

3. Estimate average second derivative of the function at the best matching unit based on the current positions of the map units.

4. Normalize this average second derivative to an interval of [0,1].

5. Move the map toward the presented data point at a rate proportional to the estimated normalizes average second derivative and iterate.

For multivariate functions only gradients along directions given by the topological structure of the map can be estimated in step 4. For example, given a 2-dimensional mesh that approximates function $f(x_1, x_2)$, every unit of the map (except the border units for which there will be only one neighbor) has two neighboring units along each topological dimension. These neighboring units can be used to approximate the function's gradients along the corresponding topological dimension of the map. These values along each dimension can then be averaged to provide a local gradient estimate at a given knot.

In step 5, estimated average second derivative $f''$ is normalized to [0,1] range using $\psi_i = 1 - \exp^{(|f''|/\tan(\tau))}$ This is done because the value of second derivative is used as the learning rate.

In step 6, the map is modified according to the following equation:

$$\mathbf{W}_j(k+1) = W_j(k) + (1 - \beta(k))\psi_i(k)C_j(k)(\mathbf{X}(k) - \mathbf{W}_j(k)) \qquad \forall j \quad (4)$$

It is this second movement of the map that allows for more flexibility around the region of the map where the second derivative is large. The process described by equation (4) is equivalent to pulling all units towards the data, with the learning rate proportional to estimated second derivative at the best matched unit. Note that the influence of the second derivative is gradually increased during the process of self-organization by the factor $(1 - \beta(k))$. This factor account for the fact that the map becomes closer and closer to the underlying function during self-organization; hence, providing a more reliable estimate of second derivative.

## 4   EMPIRICAL COMPARISON

Performance of the two algorithms (original and modified CTM) was compared for several low-dimensional problems. In all experiments the two algorithms used the same training set of 100 data points for the univariate problems and 400 data points for the 2-variable problems.

The training samples $(\mathbf{X}_i, Y_i)$ were generated according to (1), with $\mathbf{X}_i$ randomly drawn from a uniform distribution in the closed interval [-1,1], and the *error* drawn from the normal distribution $N(0, (0.1)^2)$. Regression estimates produced by the self-organized maps were tested on a different set of $n = 200$ samples (test set) generated in the same manner as the training set.

We used the Average Residual, $AR = \sqrt{\frac{1}{n} \sum_{i=1}^{n}[Y_i - f(\mathbf{X}_i)]^2}$, as the performance measure on the test set. Here, $f(X)$ is the piecewise linear estimate of the function with knot locations provided by coordinates of the units of trained CTM. The Aver-

age Residual gives an indication of standard deviation of the overall generalization error.

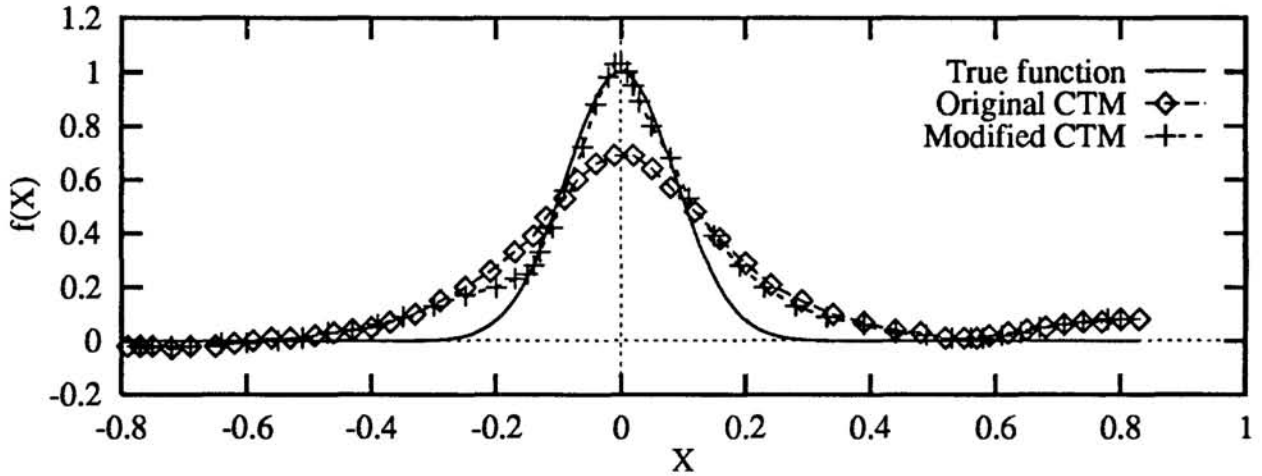

Figure 1: A 50 unit map formed by the original and modified algorithm for the Gaussian function.

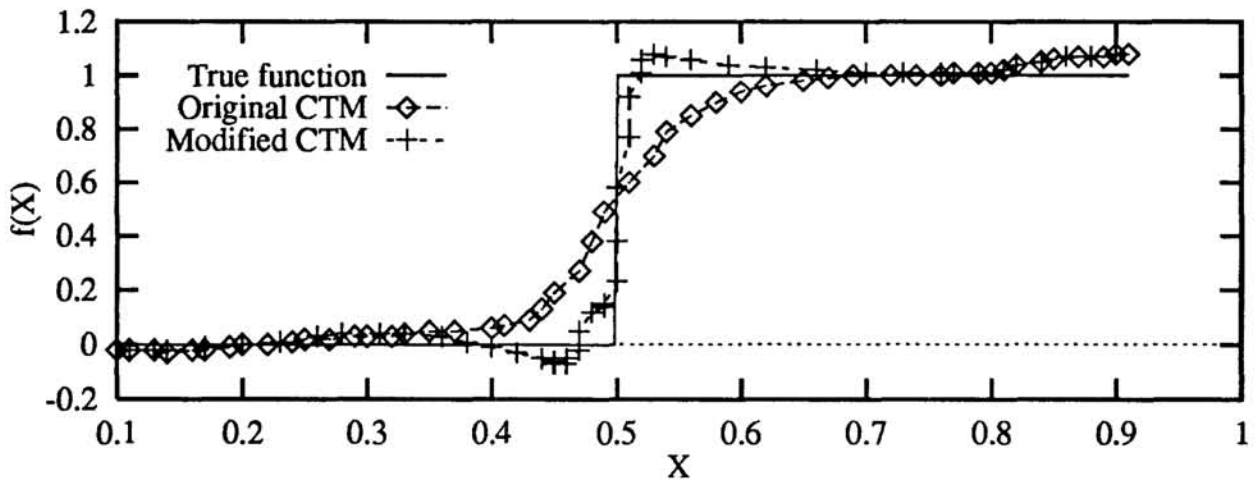

Figure 2: A 50 unit map formed by the original and modified algorithm for the step function.

We used a gaussian function ($f(x) = \exp^{-64x^2}$) and a step function for our first set of experiments. Figure 1 and 2 show the actual maps formed by the original and modified algorithm for these functions. It is clear from these figures that the modified algorithm allocates more units around the regions where the second derivative is large. This increase in the local knot density has introduced more flexibility into the model around the regions with large second derivatives. As a result of this the

model is less biased around these regions. However, there is no over-fitting in the regions where the second derivative is large.

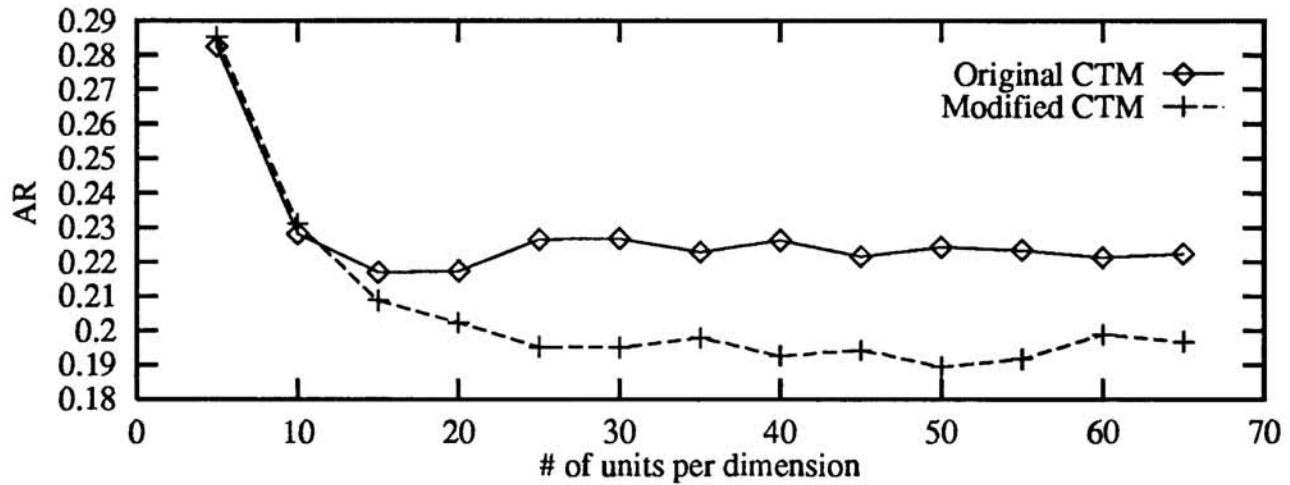

Figure 3: Average Residual error as a function of the size of the map for the 3-dimensional Step function

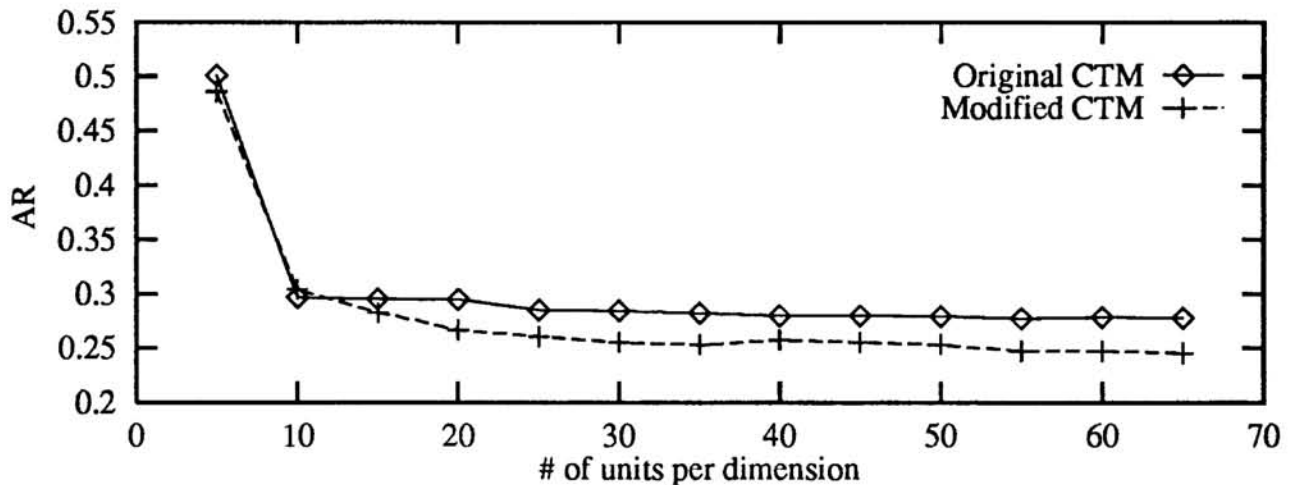

Figure 4: Average Residual error as a function of the size of the map for the 3-dimensional Sine function

To compare the behavior of the two algorithms in their predictability of structureless data, we trained them on a constant function $f(x) = 0$ with $error = N(0, (0.1)^2)$. This problem is known as smoothing pure noise in regression analysis. It has been shown [CN91] that the original algorithm handles this problem well and quality of CTM smoothing is independent of the number of units in the map. Our experiments

show that the modified algorithm performs as good as the original one in this respect.

Finally, we used the following two-variable functions (step, and sine) to see how well the modified algorithm performs in higher dimensional settings.

*Step:* $f(x_1, x_2) = \begin{cases} 1 & \text{for } ((x_1 < 0.5) \wedge (x_2 < 0.5)) \vee ((x_1 \geq 0.5) \wedge (x_2 \geq 0.5)) \\ 0 & \text{otherwise} \end{cases}$

*Sine:* $f(x_1, x_2) = \sin\left(2\pi\sqrt{(x_1)^2 + (x_2)^2}\right)$

The results of these experiments are summarized in Figure 3 and 4. Again we see that the modified algorithm outperforms the original algorithm. Note that the above example of a two-variable step function can be easily handled by recursive partitioning techniques such as CART [BFOS84]. However, recursive methods are sensitive to coordinate rotation. On the other hand, CTM is a coordinate-independent method, i.e. its performance is independent of any affine transformation in X-space.

# References

[BFOS84]  L. Breiman, J.H. Friedman, R.A. Olshen, and C.J. Stone. *Classification and Regression Trees.* Wadswordth, Belmont, CA, 1984.

[BL88]  D.S. Broomhead and D. Lowe. Multivariable functional interpolation and adaptive networks. *Complex Systems*, 2:321–355, 1988.

[Che92]  V. Cherkassky. Neural networks and nonparametric regression. In S.Y. Kung, F. Fallside, J.Aa. Sorenson, and C.A. Kamm, editors, *Neural Networks for Signal Processing*, volume II. IEEEE, Piscataway, NJ, 1992.

[CN91]  V. Cherkassky and H.L. Najafi. Constrained topological mapping for nonparametric regression analysis. *Neural Networks*, 4:27–40, 1991.

[dB78]  C. de Boor. *A Practical Guide to Splines.* Springer-Verlag, 1978.

[FS89]  J.H. Friedman and B.W. Silverman. Flexible parsimonious smoothing and additive modeling. *Technometrics*, 31(1):3–21, 1989.

[Koh84]  T. Kohonen. *Self-Organization and Associative Memory.* Springer-Verlag, third edition, 1984.

[MD89]  J. Moody and C.J. Darken. Fast learning in networks of locally tuned processing units. *Neural Computation*, 1:281, 1989.

[PG90]  T. Poggio and F. Girosi. Networks for approximation and learning. *Proceedings of the IEEE*, 78(9):1481–1497, 1990.
